# Nonconvex Penalization Using Laplace Exponents and Concave Conjugates

**Zhihua Zhang and Bojun Tu**
College of Computer Science & Technology
Zhejiang University
Hangzhou, China 310027
{zhzhang, tubojun}@zju.edu.cn

## Abstract

In this paper we study sparsity-inducing nonconvex penalty functions using Lévy processes. We define such a penalty as the Laplace exponent of a subordinator. Accordingly, we propose a novel approach for the construction of sparsity-inducing nonconvex penalties. Particularly, we show that the nonconvex logarithmic (LOG) and exponential (EXP) penalty functions are the Laplace exponents of Gamma and compound Poisson subordinators, respectively. Additionally, we explore the concave conjugate of nonconvex penalties. We find that the LOG and EXP penalties are the concave conjugates of negative Kullback-Leiber (KL) distance functions. Furthermore, the relationship between these two penalties is due to asymmetricity of the KL distance.

## 1 Introduction

Variable selection plays a fundamental role in statistical modeling for high-dimensional data sets, especially when the underlying model has a sparse representation. The approach based on penalty theory has been widely used for variable selection in the literature. A principled approach is to due the lasso of [17], which uses the $\ell_1$-norm penalty. Recently, some nonconvex alternatives, such as the bridge penalty, the nonconvex exponential penalty (EXP) [3, 8], the logarithmic penalty (LOG) [19, 13], the smoothly clipped absolute deviation (SCAD) penalty [6] and the minimax concave plus (MCP) penalty [20], have been demonstrated to have attractive properties theoretically and practically.

There has also been work on nonconvex penalties within a Bayesian framework. Zou and Li [23] derived their local linear approximation (LLA) algorithm by combining the EM algorithm with an inverse Laplace transformation. In particular, they showed that the bridge penalty can be obtained by mixing the Laplace distribution with a stable distribution. However, Zou and Li [23] proved that both MCP and SCAD can not be cast into this framework. Other authors have shown that the prior induced from the LOG penalty has an interpretation as a scale mixture of Laplace distributions with an inverse gamma density [5, 9, 12, 2]. Recently, Zhang *et al.* [22] extended this class of Laplace variance mixtures by using a generalized inverse Gaussian density. Additionally, Griffin and Brown [11] devised a family of normal-exponential-gamma priors.

Our work is motivated by recent developments of Bayesian nonparametric methods in feature selection [10, 18, 4, 15]. Especially, Polson and Scott [15] proposed a nonparametric approach for normal variance mixtures using Lévy processes, which embeds finite dimensional normal variance mixtures in infinite ones. We develop a Bayesian nonparametric approach for the construction of sparsity-inducing nonconvex penalties. Particularly, we show that Laplace transformations of Lévy processes can be viewed as pseudo-priors and the corresponding Laplace exponents then form

sparsity-inducing nonconvex penalties. Moreover, we exemplify that the LOG and EXP penalties can be respectively regarded as Laplace exponents of Gamma and compound Poisson subordinators.

In addition, we show that both LOG and EXP can be constructed via the Kullback-Leibler distance. This construction recovers an inherent connection between LOG and EXP. Moreover, it provides us with an approach for adaptively updating tuning hyperparameters, which is a very important computational issue in nonconvex sparse penalization. Typically, the multi-stage LLA and SparseNet algorithms with nonconvex penalties [21, 13] implement a two-dimensional grid research, so they take more computational costs. However, we do not claim that our method will always be optimal for generalization performance.

## 2  Lévy Processes for Nonconvex Penalty Functions

Suppose we are given a set of training data $\{(\mathbf{x}_i, y_i) : i = 1, \ldots, n\}$, where the $\mathbf{x}_i \in \mathbb{R}^p$ are the input vectors and the $y_i$ are the corresponding outputs. Moreover, we assume that $\sum_{i=1}^n \mathbf{x}_i = \mathbf{0}$ and $\sum_{i=1}^n y_i = 0$. We now consider the following linear regression model:

$$\mathbf{y} = \mathbf{X}\mathbf{b} + \varepsilon,$$

where $\mathbf{y} = (y_1, \ldots, y_n)^T$ is the $n \times 1$ output vector, $\mathbf{X} = [\mathbf{x}_1, \ldots, \mathbf{x}_n]^T$ is the $n \times p$ input matrix, and $\varepsilon$ is a Gaussian error vector $N(\varepsilon|\mathbf{0}, \sigma\mathbf{I}_n)$. We aim to find a sparse estimate of regression vector $\mathbf{b} = (b_1, \ldots, b_p)^T$ under the MAP framework.

We particular study the use of Laplace variance mixtures in sparsity modeling. For this purpose, we define a hierarchical model:

$$[b_j|\eta_j, \sigma] \overset{ind}{\sim} L(b_j|0, \sigma(2\eta_j)^{-1}),$$
$$[\eta_j] \overset{iid}{\sim} p(\eta_j),$$
$$p(\sigma) = \text{``}Constant\text{''},$$

where the $\eta_j$s are known as the local shrinkage parameters and $L(b|u, \eta)$ denotes a Laplace distribution of the density

$$L(b|u, \eta) = \frac{1}{4\eta} \exp\left(-\frac{1}{2\eta}|b - u|\right).$$

The classical regularization framework is based on a penalty function induced from the margin prior $p(b_j|\sigma)$. Let

$$\psi(|b|) = -\log p(b|\sigma),$$

where $p(b|\sigma) = \int_0^\infty L(b|0, \sigma\eta^{-1})p(\eta)d\eta$. Then the penalized regression problem is

$$\min_{\mathbf{b}} \left\{ F(b) \triangleq \frac{1}{2}\|\mathbf{y} - \mathbf{X}\mathbf{b}\|_2^2 + \lambda \sum_{j=1}^p \psi(|b_j|) \right\}.$$

Using some direct calculations, we can obtain that $\frac{d\psi(|b|)}{d|b|} > 0$ and $\frac{d^2\psi(|b|)}{d|b|^2} < 0$. This implies that $\psi(|b|)$ is nondecreasing and concave in $|b|$. In other words, $\psi(|b|)$ forms a class of nonconvex penalty functions for $b$.

Motivated by use of Bayesian nonparametrics in sparsity modeling, we now explore Laplace scale mixtures by relating $\eta$ with a subordinator. We thus have a Bayesian nonparametric formulation for the construction of joint priors of the $b_j$'s.

### 2.1  Subordinators and Laplace Exponents

Before we go into the presentation, we give some notions and lemmas that will be uses later. Let $f \in C^\infty(0, \infty)$ with $f \geq 0$. We say $f$ is completely monotone if $(-1)^n f^{(n)} \geq 0$ for all $n \in \mathbb{N}$ and a Bernstein function if $(-1)^n f^{(n)} \leq 0$ for all $n \in \mathbb{N}$. The following lemma will be useful.

**Lemma 1** *Let $\nu$ be the Lévy measure such that $\int_0^\infty \min(u, 1)\nu(du) < \infty$.*

(1) $f$ is a Bernstein function if and only if the mapping $s \mapsto \exp(-tf(s))$ is completely monotone for all $t \geq 0$.

(2) $f$ is a Bernstein function if and only if it has the representation

$$f(s) = \alpha + \beta s + \int_0^\infty \big[1 - \exp(-su)\big]\nu(du) \ \ \textit{for all } s > 0, \tag{1}$$

where $\alpha, \beta \geq 0$.

Our work is based on the notion of subordinators. Roughly speaking, a *subordinator* is an one-dimensional Lévy process that is non-decreasing (a.s.) [16]. An important property for subordinators is given in the following lemma.

**Lemma 2** *If $T = (T(t) : t \geq 0)$ is a subordinator, then the Laplace transformation of its density takes the form*

$$\mathbb{E}\big(e^{-sT(t)}\big) = \int_0^\infty e^{-sT(t)}p(T(t))dT(t) = e^{-t\psi(s)},$$

*where*

$$\psi(s) = \beta s + \int_0^\infty \big[1 - e^{-su}\big]\nu(du) \quad \textit{for } s > 0. \tag{2}$$

*Here $\beta \geq 0$ and $\nu$ is the Lévy measure defined in Lemma 1.*

*Conversely, if $\psi$ is an arbitrary mapping from $(0, \infty) \to (0, \infty)$ of the form (2), then $e^{-t\psi(s)}$ is the Laplace transformation of the density of a subordinator.*

Lemmas 1 and 2 can be found in [1, 16]. The function $\psi$ in (2) is usually called the *Laplace exponent* of the subordinator and it satisfies $\psi(0) = 0$. Lemma 1 implies that the Laplace exponent $\psi$ is a Bernstein function and the corresponding Laplace transformation $\exp(-t\psi(s))$ is completely monotone.

Recall that the Laplace exponent $\psi(s)$ is nonnegative, nondecreasing and concave on $(0, \infty)$. Thus, if we let $s = |b|$, then $\psi(|b|)$ defines a nonconvex penalty function of $b$ on $(-\infty, \infty)$. Moreover, such $\psi(|b|)$ is nondifferentiable at the origin because $\psi'(0^+) > 0$ and $\psi'(0^-) < 0$. Thus, it is able to induce sparsity. In this regard, $\exp(-t\psi(|b|))$ forms a pseudo-prior for $b$[1]. Lemma 2 shows that the prior can be defined by a Laplace transformation. In summary, we have the following theorem.

**Theorem 1** *Let $\psi(s)$ be a nonzero Bernstein function of $s$ on $(0, \infty)$. If $\psi(s) = 0$, then $\psi(|b|)$ is a nondifferentiable and nonconvex function of $b$ on $(-\infty, \infty)$. Furthermore,*

$$\exp(-t\psi(|b|)) = \int_0^\infty \exp(-|b|T(t))p(T(t))dT(t), \ t \geq 0,$$

*where $(T(t) : t \geq 0)$ is some subordinator.*

The subordinator $T(t)$ plays the same role as the local shrinkage parameter $\eta$, which is also called a latent variable. Moreover, we will see that $t$ plays the role of a tuning hyperparameter. Theorem 1 shows an explicit relationship between the local shrinkage parameter and the corresponding tuning hyperparameter; i.e., the former is a stochastic process of the later. It is also worth noting that

$$\exp(-t\psi(|b|)) = 2\int_0^\infty L(b|0, (2T(t))^{-1})T(t)^{-1}p(T(t))dT(t).$$

Thus, if $\int_0^\infty T(t)^{-1}p(T(t))dT(t) = 1/C < \infty$, $p^*(T(t)) \triangleq CT(t)^{-1}p(T(t))$ defines a new proper density for $T(t)$. In this case, the proper prior $C\exp(-t\psi(|b|))$ is a Laplace scale mixture, i.e., the mixture of $L(b|0, (2T(t))^{-1})$ with $p^*(T(t))$. If $\int_0^\infty T(t)^{-1}p(T(t))dT(t) = \infty$, then $p^*(T(t)) \triangleq T(t)^{-1}p(T(t))$ defines an improper density for $T(t)$. Thus, the improper prior $\exp(-t\psi(|b|))$ is a mixture of $L(b|0, (2T(t))^{-1})$ with $p^*(T(t))$.

## 2.2 The MAP Estimation

Based on the subordinator given in the previous subsection, we rewrite the hierarchical representation for joint prior of the $b_j$ under the regression framework. That is,

$$\begin{aligned}
[b_j|\eta_j, \sigma] &\overset{ind}{\sim} L(b_j|0, \sigma(2\eta_j)^{-1}), \\
p^*(\eta_j) &\propto \sigma\eta_j^{-1}p(\eta_j),
\end{aligned}$$

which is equivalent to

$$[b_j, \eta_j|\sigma] \overset{ind}{\propto} \exp\Big(-\frac{\eta_j}{\sigma}|b_j|\Big)p(\eta_j).$$

Here $T(t_j) = \eta_j$. The joint marginal pseudo-prior of the $b_j$'s is

$$p^*(\mathbf{b}|\sigma) = \prod_{j=1}^{p}\int_0^{\infty}\exp\Big(-\frac{\eta_j}{\sigma}|b_j|\Big)P(\eta_j)d\eta_j = \prod_{j=1}^{p}\exp\Big(-t_j\psi\Big(\frac{|b_j|}{\sigma}\Big)\Big).$$

Thus, the MAP estimate of $\mathbf{b}$ is based on the following optimization problem

$$\min_{\mathbf{b}}\ \Big\{\frac{1}{2}\|\mathbf{y} - \mathbf{Xb}\|_2^2 + \sigma\sum_{j=1}^{p}t_j\psi(|b_j|/\sigma)\Big\}.$$

Clearly, the $t_j$'s are tuning hyperparameters and the $\eta_j$'s are latent variables. Moreover, it is interesting that $\eta_j$ $(T(t_j))$ is defined as a subordinator w.r.t. $t_j$.

## 3 Gamma and Compound Poisson Subordinators

In [15], the authors discussed the use of $\alpha$-stable subordinators and inverted-beta subordinators. In this section we study applications of Gamma and Compound Poisson subordinators in constructing nonconvex penalty functions. We establish an interesting connection of these two subordinators with nonconvex logarithmic (LOG) and exponential (EXP) penalties. Particularly, these two penalties are the Laplace exponents of the two subordinators, respectively.

### 3.1 The LOG penalty and Gamma Subordinator

The log-penalty function is defined by

$$\psi(|b|) = \frac{1}{\gamma}\log\big(\alpha|b|+1\big), \quad \alpha, \gamma > 0. \tag{3}$$

Clearly, $\psi(|b|)$ is a Bernstein function of $|b|$ on $(0, \infty)$. Thus, it is the Laplace exponent of a subordinator. In particular, we have the following theorem.

**Theorem 2** *Let $\psi(s)$ be defined by (3) with $s = |b|$. Then,*

$$\frac{1}{\gamma}\log\big(\alpha s+1\big) = \int_0^{\infty}\big[1 - \exp(-su)\big]\nu(du),$$

*where the Lévy measure $\nu$ is*

$$\nu(du) = \frac{1}{\gamma u}\exp(-u/\alpha)du.$$

*Furthermore,*

$$\exp(-t\psi(s)) = (\alpha s+1)^{-t/\gamma} = \int_0^{\infty}\exp(-sT(t))p(T(t))dT(t),$$

*where $\{T(t) : t \geq 0\}$ is a Gamma subordination and each $T(t)$ has density*

$$p(T(t) = \eta) = \frac{\alpha^{-\frac{t}{\gamma}}}{\Gamma(t/\gamma)}\eta^{\frac{t}{\gamma}-1}\exp(-\alpha^{-1}\eta).$$

As we see, $T(t)$ follows Gamma distribution $\mathsf{Ga}(T(t)|t/\gamma, \alpha)$. Thus, the $\{T(t) : t \geq 0\}$ is called the Gamma subordinator.

We also note that the corresponding pseudo-prior is

$$\exp(-t\psi(|b|)) = \left(\alpha|b|+1\right)^{-t/\gamma} \propto \int_0^\infty L(b|0, T(t)^{-1})T(t)^{-1}p(T(t))dT(t).$$

Furthermore, if $t > \gamma$, we can form the pseudo-prior as a proper distribution, which is the mixture of $L(b|0, T(t)^{-1})$ with Gamma distribution $\mathsf{Ga}(T(t)|\gamma^{-1}t-1, \alpha)$.

## 3.2 The EXP Penalty and Compound Poisson Subordinator

We call $\{K(t), t \geq 0\}$ a Poisson process of intensity $\lambda > 0$ if $K$ takes values in $\mathbb{N} \cup \{0\}$ and each $K(t) \sim \mathsf{Po}(K(t)|\lambda t)$, namely,

$$P(K(t) = k) = \frac{(\lambda t)^k}{k!}e^{-\lambda t}, \text{ for } k = 0, 1, 2, \ldots$$

Let $\{Z(k) : k \in \mathbb{N}\}$ be a sequence of i.i.d. random real variables from common law $\mu_Z$ and let $K$ be a Poisson process of intensity $\lambda$ that is independent of all the $Z(k)$. Then $T(t) \triangleq Z(K(1)) + \cdots + Z(K(t))$ for $t \geq 0$ follows a compound Poisson distribution (denoted $T(t) \sim \mathsf{Po}(T(t)|\lambda t, \mu_Z)$). We then call $\{T(t) : t \geq 0\}$ the compound Poisson process. It is well known that Poisson processes are subordinators. A compound Poisson process is a subordinator if and only if the $Z(k)$ are nonnegative random variables [16].

In this section we employ the compound Poisson process to explore the EXP penalty, which is

$$\psi(|b|) = \frac{1}{\gamma}(1 - \exp(-\alpha|b|)), \quad \alpha, \gamma > 0. \tag{4}$$

It is easily seen that $\psi(|b|)$ is a Bernstein function of $|b|$ on $(0, \infty)$. Moreover, we have

**Theorem 3** *Let $\psi(s)$ be defined by (4) where $|b| = s$. Then*

$$\psi(s) = \int_0^\infty [1 - \exp(-su)]\nu(du)$$

*with the Lévy measure $\nu(du) = \gamma^{-1}\delta_\alpha(u)du$. Furthermore,*

$$\exp(-t\psi(s)) = \int_0^\infty \exp(-sT(t))P(T(t))dT(t),$$

*where $\{T(t) : t \geq 0\}$ is a compound Poisson subordinator, each $T(t) \sim \mathsf{Po}(T(t)|t/\gamma, \delta_\alpha(\cdot))$, and $\delta_u(\cdot)$ is the Dirac Delta measure.*

Note that $\int_\mathbb{R} (1 - \exp(-\alpha|b|))db = \infty$, so $\gamma^{-1}(1 - \exp(-\alpha|b|))$ is an improper prior of $b$.

As we see, there are two parameters $\alpha$ and $\gamma$ in both LOG and EXP penalties. Usually, for the LOG penalty ones set $\gamma = \log(1 + \alpha)$, because the corresponding $\psi(|b|)$ goes from $\|b\|_1$ to $\|b\|_0$, as $\alpha$ varying from 0 to $\infty$. In the same reason, ones set $\gamma = 1 - \exp(-\alpha)$ for the EXP penalty. Thus, $\alpha$ (or $\gamma$) measures the sparseness. It makes sense to set $\alpha$ as $\alpha = p$ (i.e., the dimension of the input vector) in the following experiments. Interestingly, the following theorem shows a limiting property of the subordinators.

**Theorem 4** *Assume that $\alpha > 0$ and $\gamma > 0$.*

(1) *If $\gamma = \log(1 + \alpha)$, then $\lim_{\alpha \to 0} \mathsf{Ga}(T(t)|t/\gamma, \alpha) \xrightarrow{d} \delta_t(T(t))$.*

(2) *If $\gamma = 1 - e^{-\alpha}$, then $\lim_{\alpha \to 0} \mathsf{Po}(T(t)|t/\gamma, \delta_\alpha(\cdot)) \xrightarrow{d} \delta_t(T(t))$.*

In this section we have an interesting connection between the LOG and EXP penalties based on the relationship between the Gamma and compound Poisson subordinators. Subordinators help

us establish a direct connection between the tuning hyperparameters $t_j$ and the latent variables $\eta_j$ ($T(t_j)$). However, when we implement the MAP estimation, it is challenging how to select these tuning hyperparameters. Recently, Palmer *et al.* [14] considered the application of concave conjugates in developing variational EM algorithms for non-Gaussian latent variable models. In the next section we rederive the nonconvex LOG and EXP penalties via concave conjugate. This derivation is able to deal with the challenge.

## 4 A View of Concave Conjugate

Our derivation for the LOG and EXP penalties is based on the Kullback-Leibler (KL) distance. Given two nonnegative vectors $\mathbf{a} = (a_1, \ldots, a_p)^T$ and $\mathbf{s} = (s_1, \ldots, s_p)^T$, the KL distance between them is

$$\mathrm{KL}(\mathbf{a}, \mathbf{s}) = \sum_{j=1}^{p} a_j \log \frac{a_j}{s_j} - a_j + s_j,$$

where $0 \log \frac{0}{0} = 0$. It is well known that $\mathrm{KL}(\mathbf{a}, \mathbf{s}) \geq 0$ and $\mathrm{KL}(\mathbf{a}, \mathbf{s}) = 0$ if and only if $\mathbf{a} = \mathbf{s}$, but typically $\mathrm{KL}(\mathbf{a}, \mathbf{s}) \neq \mathrm{KL}(\mathbf{s}, \mathbf{a})$.

**Theorem 5** *Let $\mathbf{a} = (a_1, \ldots, a_p)^T$ be a nonnegative vector and $|\mathbf{b}| = (|b_1|, \ldots, |b_p|)^T$. Then,*

$$\sum_{j=1}^{p} a_j \psi(|b_j|) \triangleq \sum_{j=1}^{p} \frac{a_j}{\alpha} \log\left(\alpha|b_j|+1\right) = \min_{\mathbf{w} \geq 0} \left\{ \mathbf{w}^T |\mathbf{b}| + \frac{1}{\alpha} \mathrm{KL}(\mathbf{a}, \mathbf{w}) \right\}$$

*when $w_j = a_j/(1 + \alpha|b_j|)$, and*

$$\sum_{j=1}^{p} a_j \psi(|b_j|) \triangleq \sum_{j=1}^{p} \frac{a_j}{\alpha} [1 - \exp(-\alpha|b_j|)] = \min_{\mathbf{w} \geq 0} \left\{ \mathbf{w}^T |\mathbf{b}| + \frac{1}{\alpha} \mathrm{KL}(\mathbf{w}, \mathbf{a}) \right\}$$

*when $w_j = a_j \exp(-\alpha|b_j|)$.*

When setting $a_j = \frac{\alpha}{\gamma} t_j$, we readily see the LOG and EXP penalties. Thus, Theorem 5 illustrates a very interesting connection between the LOG and EXP penalties. Since $\mathrm{KL}(\mathbf{a}, \mathbf{w})$ is strictly convex in either $\mathbf{w}$ or $\mathbf{a}$, the LOG and EXP penalties are respectively the concave conjugates of $-\alpha^{-1}\mathrm{KL}(\mathbf{a}, \mathbf{w})$ and $-\alpha^{-1}\mathrm{KL}(\mathbf{w}, \mathbf{a})$.

The construction method for the nonconvex penalties provides us with a new approach for solving the corresponding penalized regression model. In particular, to solve the nonconvex penalized regression problem:

$$\min_{\mathbf{b}} \left\{ J(\mathbf{b}, \mathbf{a}) \triangleq \frac{1}{2} \|\mathbf{y} - \mathbf{X}\mathbf{b}\|_2^2 + \sum_{j=1}^{p} a_j \psi(|b_j|) \right\}, \tag{5}$$

we equivalently formulate it as

$$\min_{\mathbf{b}} \left\{ \min_{\mathbf{w} \geq \mathbf{0}} \left\{ \frac{1}{2} \|\mathbf{y} - \mathbf{X}\mathbf{b}\|_2^2 + \mathbf{w}^T |\mathbf{b}| + \frac{1}{\alpha} D(\mathbf{w}, \mathbf{a}) \right\} \right\}. \tag{6}$$

Here $D(\mathbf{w}, \mathbf{a})$ is either $\mathrm{KL}(\mathbf{a}, \mathbf{w})$ or $\mathrm{KL}(\mathbf{w}, \mathbf{a})$. Moreover, we are also interested in adaptive estimation of $\mathbf{a}$ in solving the problem (6). Accordingly, we develop a new training algorithm, which consists of two steps.

We are given initial values $\mathbf{w}^{(0)}$, e.g., $\mathbf{w}^{(0)} = (1, \ldots, 1)^T$. After the $k$th estimates $(\mathbf{b}^{(k)}, \mathbf{a}^{(k)})$ of $(\mathbf{b}, \mathbf{a})$ are obtained, the $(k+1)$th iteration of the algorithm is defined as follows.

The first step calculates $\mathbf{w}^{(k)}$ via

$$\mathbf{w}^{(k)} = \operatorname*{argmin}_{\mathbf{w} > 0} \left\{ \sum_{j=1}^{p} w_j |b_j^{(k)}| + \frac{1}{\alpha} D(\mathbf{w}, \mathbf{a}^{(k)}) \right\}.$$

Particular, $w_j^{(k)} = a_j^{(k)}/(1 + \alpha|b_j^{(k)}|)$ in LOG, while $w_j^{(k)} = a_j^{(k)} \exp(-\alpha|b_j^{(k)}|)$ in EXP.

The second step then calculates $(\mathbf{b}^{(k+1)}, \mathbf{a}^{(k+1)})$ via

$$(\mathbf{b}^{(k+1)}, \mathbf{a}^{(k+1)}) = \underset{\mathbf{b}, \mathbf{a}}{\operatorname{argmin}} \left\{ \frac{1}{2}\|\mathbf{y} - \mathbf{Xb}\|_2^2 + |\mathbf{b}|^T\mathbf{w}^{(k)} + \frac{1}{\alpha}D(\mathbf{w}^{(k)}, \mathbf{a}) \right\}.$$

Note that given $\mathbf{w}^{(k)}$, $\mathbf{b}$ and $\mathbf{a}$ are independent. Thus, this step can be partitioned into two parts. Namely, $\mathbf{a}^{(k+1)} = \mathbf{w}^{(k)}$ and

$$\mathbf{b}^{(k+1)} = \underset{\mathbf{b}}{\operatorname{argmin}} \left\{ \frac{1}{2}\|\mathbf{y} - \mathbf{Xb}\|_2^2 + \sum_{j=1}^{p} w_j^{(k)}|b_j| \right\}.$$

Recall that the LOG and EXP penalties are differentiable and strictly concave in $|b|$ on $[0, \infty)$. Thus, the above algorithm enjoys the same convergence property of the LLA was studied by Zou and Li [23] (see Theorem 1 and Proposition 1 therein).

## 5 Experimental Analysis

We conduct experimental analysis of our algorithms with LOG and EXP given in the previous section. We also implement the Lasso, adaptive Lasso (adLasso) and MCP-based methods. All these methods are solved by the coordinate descent algorithm. For LOD and EXP algorithms, we fix $\alpha = p$ (the dimension of the input vector), and set $\mathbf{w}^{(0)} = \omega\mathbf{1}$ where $\omega$ is selected by using cross-validation and $\mathbf{1}$ is the vector of ones. For Lasso, AdLasso and MCP, we use cross-validation to select the tunning parameters ($\lambda$ in Lasso, $\lambda$ and $\gamma$ in AdLasso and MCP).

In this simulation example, we use a data model as follow

$$y = \mathbf{x}^T\mathbf{b} + \sigma\epsilon$$

where $\epsilon \sim N(0, 1)$, and $\mathbf{b}$ is a 200-dimension vector with only 10 non-zeros such that $b_i = b_{100+i} = 0.2i$, $i = 1, \ldots, 5$. Each data point $\mathbf{x}$ is sampled from a multivariate normal distribution with zero mean and covariance matrix $\Sigma = \{0.7^{|i-j|}\}_{1 \leq i,j \leq 200}$. We choose $\sigma$ such that the Signal-to-Noise Ratio (SNR), which is defined as

$$\text{SNR} = \frac{\sqrt{\mathbf{b}^T\Sigma\mathbf{b}}}{\sigma},$$

is a specified value. Our experiment is performed on $n = 100$ and two different SNR values. We generate $N = 1000$ test data for each test. Let $\hat{\mathbf{b}}$ denote the solution given by each algorithm. The Standardized Prediction Error (SPE) is defined as

$$\text{SPE} = \frac{\sum_{i=1}^{N}(y_i - \mathbf{x}_i^T\hat{\mathbf{b}})^2}{N\sigma^2}$$

and the Feature Selection Error (FSE) is proportion of coefficients in $\hat{\mathbf{b}}$ which is correctly set to zero or non-zero based on true $\mathbf{b}$.

Figure 1 reports the average results over 20 repeats. From the figure, we see that both the LOG and EXP outperform the other methods in prediction accuracy and sparseness in most cases. Our methods usually takes about 10 iterations to get convergence. Thus, our methods are computationally more efficient than the AdLasso and MCP.

In the second experiment, we apply our methods to regression problems on four datasets from UCI Machine Learning Repository and the cookie (Near-Infrared (NIR) Spectroscopy of Biscuit Doughs) dataset [7]. For the four UCI datasets, we randomly select 70% of the data for training and the rest for test, and repeat this process for 20 times. We report the mean and standard deviation of the Root Mean Square Error (RMSE) and the model sparsity (proportion of zero coefficients in the model) in Tables 1 and 2. For the NIR dataset, we follow the steup for the original dataset: 40 instances for training and 32 instances for test. We form four different datasets for the four responses ("fat", "sucrose", "dry flour" and "water") in the experiment, and report the RMSE on the test set and the model sparsity in Table 3. We can see that all the methods are competitive in both prediction accuracy. But the nonconvex LOG, EXP and MCP have strong ability in feature selection.

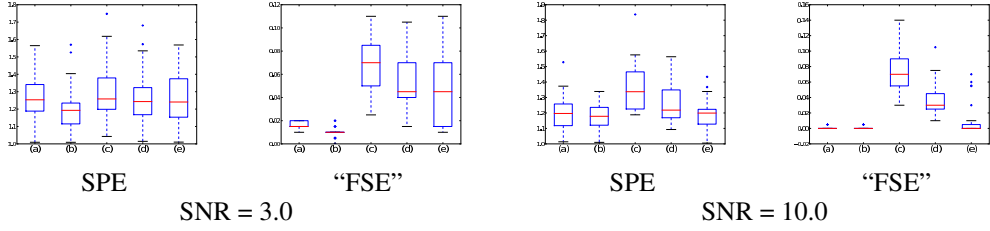

Figure 1: Box-and-whisker plots of SPE and FSE results. Here (a), (b), (c), (d), (e) are for LOG, EXP, Lasso, AdLasso, and MCP, respectively

Table 1: Root Mean Square Error on Real datasets

|         | Abalone          | Housing          | Pyrim            | Triazines        |
|---------|------------------|------------------|------------------|------------------|
| LOG     | **2.207**(±0.077) | **4.880**(±0.405) | 0.138(±0.032)   | 0.156(±0.018)   |
| EXP     | 2.208(±0.077)    | 4.883(±0.405)    | 0.130(±0.033)   | 0.153(±0.020)   |
| Lasso   | 2.208(±0.078)    | 4.886(±0.414)    | **0.118**(±0.035) | **0.146**(±0.017) |
| AdLasso | 2.208(±0.078)    | 4.887(±0.413)    | 0.127(±0.028)   | 0.146(±0.017)   |
| MCP     | 2.209(±0.078)    | 4.889(±0.412)    | 0.122(±0.036)   | 0.148(±0.017)   |

Table 2: Sparsity on Real datasets

|         | Abalone          | Housing          | Pyrim            | Triazines        |
|---------|------------------|------------------|------------------|------------------|
| LOG     | **12.50**(±0.00) | **11.54**(±5.70) | 57.22(±35.32)   | 68.17(±31.19)   |
| EXP     | 10.63(±4.46)     | 8.08(±5.15)      | **88.15**(±5.69) | **76.25**(±21.84) |
| Lasso   | 1.88(±4.46)      | 3.08(±5.10)      | 36.48(±24.52)   | 62.08(±14.65)   |
| AdLasso | 8.75(±5.73)      | 8.07(±7.08)      | 34.62(±28.81)   | 63.58(±15.18)   |
| MCP     | **12.50**(±0.00) | **11.54**(±6.66) | 41.48(±23.88)   | 73.00(±18.77)   |

Table 3: Root Mean Square Error and Sparsity on Real datasets NIR

|         | NIR(fat) | | NIR(sucrose) | | NIR(dry flour) | | NIR(water) | |
|---------|-------|----------|-------|----------|-------|----------|-------|----------|
|         | RMSE  | Sparsity | RMSE  | Sparsity | RMSE  | Sparsity | RMSE  | Sparsity |
| LOG     | 0.334 | **99.14** | **1.45** | **98.71** | 0.992 | **99.71** | 0.400 | **98.14** |
| EXP     | **0.307** | 97.29 | 1.47  | 97.71    | 0.908 | 98.86    | 0.484 | 94.14    |
| Lasso   | 0.437 | 68.86    | 2.54  | 53.43    | **0.785** | 92.29 | **0.378** | 65.57 |
| AdLasso | 0.835 | 88.14    | 2.22  | 86.14    | 0.862 | 99.14    | 0.407 | 85.86    |
| MCP     | 0.943 | 94.14    | 2.07  | 95.43    | 0.839 | **99.71** | 0.504 | 96.29   |

# 6  Conclusion

In this paper we have introduced subordinators of Lévy processes into the definition of nonconvex penalties. This leads us to a Bayesian nonparametric approach for constructing sparsity-inducing penalties. In particular, we have illustrated the construction of the LOG and EXP penalties. Along this line, it would be interesting to investigate other penalty functions via subordinators and compare the performance of these penalties. We will conduct a comprehensive study in the future work.

## Acknowledgments

This work has been supported in part by the Natural Science Foundations of China (No. 61070239).

## Footnotes

[1]If $\int_0^\infty \exp(-t\psi(s))ds$ is infinite, $\exp(-t\psi(|b|))$ is an improper density w.r.t. Lebesgue measure. Otherwise, it can forms a proper density. In any case, we use the terminology of pseudo-priors for $\exp(-t\psi(|b|))$.

# References

[1] D. Applebaum. *Lévy Processes and Stochastic Calculus*. Cambridge University Press, Cambridge, UK, 2004.

[2] A. Armagan, D. Dunson, and J. Lee. Generalized double Pareto shrinkage. Technical report, Duke University Department of Statistical Science, February 2011.

[3] P. S. Bradley and O. L. Mangasarian. Feature selection via concave minimization and support vector machines. In *The 26th International Conference on Machine Learning*, pages 82–90. Morgan Kaufmann Publishers, San Francisco, California, 1998.

[4] F. Caron and A. Doucet. Sparse bayesian nonparametric regression. In *Proceedings of the 25th international conference on Machine learning*, page 88, 2008.

[5] V. Cevher. Learning with compressible priors. In *Advances in Neural Information Processing Systems 22*, pages 261–269, 2009.

[6] J. Fan and R. Li. Variable selection via nonconcave penalized likelihood and its Oracle properties. *Journal of the American Statistical Association*, 96:1348–1361, 2001.

[7] Osborne B. G., Fearn T., Miller A. R., and Douglas S. Application of near-infrared reflectance spectroscopy to compositional analysis of biscuits and biscuit dough. *Journal of the Science of Food and Agriculture*, 35(1):99–105, 1984.

[8] C. Gao, N. Wang, Q. Yu, and Z. Zhang. A feasible nonconvex relaxation approach to feature selection. In *Proceedings of the Twenty-Fifth National Conference on Artificial Intelligence (AAAI'11)*, 2011.

[9] P. J. Garrigues and B. A. Olshausen. Group sparse coding with a Laplacian scale mixture prior. In *Advances in Neural Information Processing Systems 22*, 2010.

[10] Z. Ghahramani, T. Griffiths, and P. Sollich. Bayesian nonparametric latent feature models. In *World meeting on Bayesian Statistics*, 2006.

[11] J. E. Griffin and P. J. Brown. Bayesian adaptive Lassos with non-convex penalization. Technical report, University of Kent, 2010.

[12] A. Lee, F. Caron, A. Doucet, and C. Holmes. A hierarchical Bayesian framework for constructing sparsity-inducing priors. Technical report, University of Oxford, UK, 2010.

[13] R. Mazumder, J. Friedman, and T. Hastie. SparseNet: Coordinate descent with nonconvex penalties. *Journal of the American Statistical Association*, 106(495):1125–1138, 2011.

[14] J. A. Palmer, D. P. Wipf, K. Kreutz-Delgado, and B. D. Rao. Variational EM algorithms for non-Gaussian latent variable models. In *Advances in Neural Information Processing Systems 18*, 2006.

[15] N. G. Polson and J. G. Scott. Local shrinkage rules, lévy processes, and regularized regression. *Journal of the Royal Statistical Society (Series B)*, 74(2):287–311, 2012.

[16] S.-I. P. Sato. *Lévy Processes and infinitely Divisible Distributions*. Cambridge University Press, Cambridge, UK, 1999.

[17] R. Tibshirani. Regression shrinkage and selection via the lasso. *Journal of the Royal Statistical Society, Series B*, 58:267–288, 1996.

[18] M. K. Titsias. The infinite gamma-poisson feature models. In *Advances in Neural Information Processing Systems 20*, 2007.

[19] J. Weston, A. Elisseeff, B. Schölkopf, and M. Tipping. Use of the zero-norm with linear models and kernel methods. *Journal of Machine Learning Research*, 3:1439–1461, 2003.

[20] C.-H. Zhang. Nearly unbiased variable selection under minimax concave penalty. *The Annals of Statistics*, 38:894–942, 2010.

[21] T. Zhang. Analysis of multi-stage convex relaxation for sparse regularization. *Journal of Machine Learning Research*, 11:1081–1107, 2010.

[22] Z. Zhang, S. Wang, D. Liu, and M. I. Jordan. EP-GIG priors and applications in Bayesian sparse learning. *Journal of Machine Learning Research*, 13:2031–2061, 2012.

[23] H. Zou and R. Li. One-step sparse estimates in nonconcave penalized likelihood models. *The Annals of Statistics*, 36(4):1509–1533, 2008.

